# On Herding and the Perceptron Cycling Theorem

**Andrew E. Gelfand, Yutian Chen, Max Welling**
Department of Computer Science
University of California, Irvine
{agelfand,yutianc,welling}@ics.uci.edu

**Laurens van der Maaten**
Department of CSE, UC San Diego
PRB Lab, Delft University of Tech.
lvdmaaten@gmail.com

## Abstract

The paper develops a connection between traditional perceptron algorithms and recently introduced herding algorithms. It is shown that both algorithms can be viewed as an application of the perceptron cycling theorem. This connection strengthens some herding results and suggests new (supervised) herding algorithms that, like CRFs or discriminative RBMs, make predictions by conditioning on the input attributes. We develop and investigate variants of conditional herding, and show that conditional herding leads to practical algorithms that perform better than or on par with related classifiers such as the voted perceptron and the discriminative RBM.

## 1 Introduction

The invention of the perceptron [12] goes back to the very beginning of AI more than half a century ago. Rosenblatt's very simple, neurally plausible learning rule made it an attractive algorithm for learning relations in data: for every input $\mathbf{x}_i$, make a linear prediction about its label: $y_i^* = \mathbf{w}^T \mathbf{x}_i$ and update the weights as,

$$\mathbf{w} \leftarrow \mathbf{w} + \mathbf{x}_i(y_i - y_i^*) \tag{1}$$

A critical evaluation by Minsky and Papert [11] revealed the perceptron's limited representational power. This fact is reflected in the behavior of Rosenblatt's learning rule: if the data is linearly separable, then the learning rule converges to the correct solution in a number of iterations that can be bounded by $(R/\gamma)^2$, where $R$ represents the norm of the largest input vector and $\gamma$ represents the margin between the decision boundary and the closest data-case. However, '*for data sets that are not linearly separable, the perceptron learning algorithm will never converge*' (quoted from [1]).

While the above result is true, the theorem in question has something much more powerful to say. The '*perceptron cycling theorem*' (PCT) [2, 11] states that for the inseparable case the weights remain bounded and do not diverge to infinity. In this paper, we show that the implication of this theorem is that certain moments are conserved on average. Denoting the data-case selected at iteration $t$ by $i_t$ (note that the same data-case can be picked multiple times), the corresponding attribute vector and label by $(\mathbf{x}_{i_t}, y_{i_t})$ with $\mathbf{x}_i \in \mathcal{X}$, and the label predicted by the perceptron at iteration $t$ for data-case $i_t$ by $y_{i_t}^*$, we obtain the following result:

$$||\frac{1}{T}\sum_{t=1}^{T} \mathbf{x}_{i_t} y_{i_t} - \frac{1}{T}\sum_{t=1}^{T} \mathbf{x}_{i_t} y_{i_t}^*|| \sim \mathcal{O}(1/T) \tag{2}$$

This result implies that, even though the perceptron learning algorithm does not converge in the inseparable case, it generates predictions that correlate with the attributes in the same way as the true labels do. More importantly, the correlations converge to the sample mean with a rate $1/T$, which is much faster than sampling based algorithms that converge at a rate $1/\sqrt{T}$. By using general features $\phi(\mathbf{x})$, the above result can be extended to the matching of arbitrarily complicated statistics between data and predictions.

In the inseparable case, we can interpret the perceptron as a bagging procedure and average predictions instead of picking the single best (or last) weights found during training. Although not directly motivated by the PCT and Eqn. 2, this is exactly what the *voted perceptron* (VP) [5] does. Interesting generalization bounds for the voted perceptron have been derived in [5]. Extensions of VP to chain models have been explored in, e.g. [4].

*Herding* is a seemingly unrelated family of algorithms for unsupervised learning [15, 14, 16, 3]. In traditional methods for learning Markov Random Field (MRF) models, the goal is to converge to a single parameter estimate and then perform (approximate) inference in the resulting model. In contrast, herding combines the learning and inference phases by treating the weights as dynamic quantities and defining a deterministic set of updates such that averaging predictions preserves certain moments of the training data. The herding algorithm generates a weakly chaotic sequence of weights and a sequence of states of both hidden and visible variables of the MRF model. The intermediate states produced by herding are really 'representative points' of an implicit model that interpolates between data cases. We can view these states as pseudo-samples, which analogously to Eqn. 2, satisfy certain constraints on their average sufficient statistics. However, unlike in perceptron learning, the non-convergence of the weights is needed to generate long, non-periodic trajectories of states that can be averaged over.

In this paper, we show that *supervised* perceptron algorithms and *unsupervised* herding algorithms can all be derived from the PCT. This connection allows us to strengthen existing herding results. For instance, we prove fast convergence rates of sample averages when we use small mini-batches for making updates, or when we use incomplete optimization algorithms to run herding. Moreover, the connection suggests new algorithms that lie between supervised perceptron and unsupervised herding algorithms. We refer to these algorithms as "*conditional herding*" (CH) because, like conditional random fields, they condition on the input features. From the perceptron perspective, conditional herding can be understood as "*voted perceptrons with hidden units*". Conditional herding can also be interpreted as the zero temperature limit of discriminative RBMs (dRBMs) [10].

## 2   Perceptrons, Herding and the Perceptron Cycling Theorem

We first review the perceptron cycling theorem that was initially introduced in [11] with a gap in the proof that was fixed in [2]. A sequence of vectors $\{\mathbf{w}_t\}, \mathbf{w}_t \in \mathbb{R}^D, t = 0, 1, \dots$ is generated by the following iterative procedure: $\mathbf{w}_{t+1} = \mathbf{w}_t + \mathbf{v}_t$, where $\mathbf{v}_t$ is an element of a *finite set*, $\mathbf{V}$, and the norm of $\mathbf{v}_t$ is bounded: $\max_i ||\mathbf{v}_i|| = R < \infty$.

**Perceptron Cycling Theorem (PCT).** $\forall t \geq 0$: *If* $\mathbf{w}_t^T \mathbf{v}_t \leq 0$, *then there exists a constant* $M > 0$ *such that* $||\mathbf{w}_t - \mathbf{w}_0|| < M$.

The theorem still holds when $\mathbf{V}$ is a finite set in a Hilbert space. The PCT immediately leads to the following result:

**Convergence Theorem.** *If PCT holds, then:* $||\frac{1}{T} \sum_{t=1}^{T} \mathbf{v}_t|| \sim \mathcal{O}(1/T)$.

This result is easily shown by observing that $||\mathbf{w}_{T+1} - \mathbf{w}_0|| = ||\sum_{t=1}^{T} \Delta\mathbf{w}_t|| = ||\sum_{t=1}^{T} \mathbf{v}_t|| < M$, and dividing all terms by $T$.

### 2.1   Voted Perceptron and Moment Matching

The voted perceptron (VP) algorithm [5] repeatedly applies the update rule in Eqn. 1. Predictions of test labels are made after each update and final label predictions are taken as an average of all intermediate predictions. The PCT convergence theorem leads to the result of Eqn. 2, where we identify $\mathbf{V} = \{\mathbf{x}_i(y_i - y_i^*)\}, y_i = \pm 1, y_i^* = \pm 1, i = 1, \dots, N\}$. For the VP algorithm, the PCT thus guarantees that the moments $\langle \mathbf{x}y \rangle_{\tilde{p}(\mathbf{x}, y)}$ (with $\tilde{p}$ the empirical distribution) are matched with $\langle \mathbf{x}y^* \rangle_{p(y^*|\mathbf{x})\tilde{p}(\mathbf{x})}$ where $p(y^*|\mathbf{x})$ is the model distribution implied by how VP generates $y^*$.

In maximum entropy models, one seeks a model that satisfies a set of expectation constraints (moments) from the training data, while maximizing the entropy of the remaining degrees of freedom [9]. In contrast, a single perceptron strives to learn a deterministic mapping $p(y^*|\mathbf{x}) = \delta[y^* - \arg\max_y (y\mathbf{w}^T\mathbf{x})]$ that has zero entropy and gets every prediction on every training case

correct (where $\delta$ is the delta function). Entropy is created in $p(y^*|\mathbf{x})$ only when the weights $\mathbf{w}_t$ do not converge (i.e. for inseparable data sets). Thus, VP and maximum entropy methods are related, but differ in how they handle the degrees of freedom that are unconstrained by moment matching.

## 2.2 Herding

A new class of unsupervised learning algorithms, known as "herding", was introduced in [15]. Rather than learning a single 'best' MRF model that can be sampled from to estimate quantities of interest, herding combines learning and inference into a single process. In particular, herding produces a trajectory of weights and states that reproduce the moments of the training data.

Consider a fully observed MRF with features $\phi(\mathbf{x})$, $\mathbf{x} \in \mathcal{X} = [1, \ldots, K]^m$ with $K$ the number of states for each variable $x_j$ ($j = 1, \ldots, m$) and with an energy function $E(\mathbf{x})$ given by:

$$E(\mathbf{x}) = -\mathbf{w}^T \phi(\mathbf{x}). \tag{3}$$

In herding [15], the parameters $\mathbf{w}$ are updated as:

$$\mathbf{w}_{t+1} = \mathbf{w}_t + \overline{\phi} - \phi(\mathbf{x}_t^*), \tag{4}$$

where $\overline{\phi} = \frac{1}{N}\sum_i \phi(\mathbf{x}_i)$ and $\mathbf{x}_t^* = \arg\max_{\mathbf{x}} \mathbf{w}_t^T \phi(\mathbf{x})$. Eqn. 4 looks like a maximum likelihood (ML) gradient update, with constant learning rate and maximization in place of expectation in the right-hand side. This follows from taking the zero temperature limit of the ML objective (see Section 2.5). The maximization prevents the herding sequence from converging to a single point estimate on this alternative objective.

Let $\{\mathbf{w}_t\}$ denote the sequence of weights and $\{\mathbf{x}_t^*\}$ denote the sequence of states (pseudo-samples) produced by herding. We can apply the PCT to herding by identifying $\mathbf{V} = \{\overline{\phi} - \phi(\mathbf{x}^*)|\ \mathbf{x}^* \in \mathcal{X}\}$. It is now easy to see that, in general, herding does not converge because under very mild conditions we can always find an $\mathbf{x}_t^*$ such that $\mathbf{w}_t^T \mathbf{v}_t < 0$. From the PCT convergence theorem, we also see that $||\overline{\phi} - \frac{1}{T}\sum_{t=1}^T \phi(\mathbf{x}_t^*)|| \sim \mathcal{O}(1/T)$, i.e. the pseudo-sample averages of the features converge to the data averages $\overline{\phi}$ at a rate $1/T$ [1]. This is considerably faster than i.i.d. sampling from the corresponding MRF model, which would converge at a rate of $1/\sqrt{T}$.

Since the cardinality of the set $\mathbf{V}$ is exponentially large (i.e. $|\mathbf{V}| = K^m$), finding the maximizing state $\mathbf{x}_t^*$ at each update may be hard. However, the PCT only requires us to find *some* state $\mathbf{x}_t^*$ such that $\mathbf{w}_t^T \mathbf{v}_t \leq 0$ and in most cases this can easily be verified. Hence, the PCT provides a theoretical justification for using a local search algorithm that performs partial energy maximization. For example, we may start the local search from the state we ended up in during the previous iteration (a so-called persistent chain [13, 17]). Or, one may consider *contrastive divergence*-like algorithms [8], in which the sampling or mean field approximation is replaced by a maximization. In this case, maximizations are initialized on all data-cases and the weights are updated by the difference between the average over the data-cases minus the average over the $\{\mathbf{x}_i^*\}$ found after (partial) maximization. In this case, the set $\mathbf{V}$ is given by: $\mathbf{V} = \{\overline{\phi} - \frac{1}{N}\sum_i \phi(\mathbf{x}_i^*)|\ \mathbf{x}_i^* \in \mathcal{X}\ \forall i\}$. For obvious reasons, it is now guaranteed that $\mathbf{w}_t^T \mathbf{v}_t \leq 0$.

In practice, we often use mini-batches of size $n < N$ instead of the full data set. In this case, the cardinality of the set $\mathbf{V}$ is enlarged to $|\mathbf{V}| = C(n, N)K^m$, with $C(n, N)$ representing the 'n choose N' ways to compute the sample mean $\overline{\phi}_{(n)}$ based on a subset of $n$ data-cases. The negative term remains unaltered. Since the PCT still applies: $||\frac{1}{T}\sum_{t=1}^T \overline{\phi}_{(n),t} - \frac{1}{T}\sum_{t=1}^T \phi(\mathbf{x}_t^*)|| \sim \mathcal{O}(1/T)$. Depending on how the mini-batches are picked, convergence onto the overall mean $\overline{\phi}$ can be either $\mathcal{O}(1/\sqrt{T})$ (random sampling with replacement) or $\mathcal{O}(1/T)$ (sampling without replacement which has picked all data-cases after $\lceil N/n \rceil$ rounds).

## 2.3 Hidden Variables

The discussion so far has considered only constant features: $\phi(\mathbf{x}, y) = \mathbf{x}y$ for VP and $\phi(\mathbf{x})$ for herding. However, the PCT allows us to consider more general features that depend on the weights

$\mathbf{w}$, as long as the image of this feature mapping (and therefore, the update vector $\mathbf{v}$) is a set of finite cardinality. In [14], such features took the form of 'hidden units':

$$\phi(\mathbf{x},\mathbf{z}), \qquad \mathbf{z}(\mathbf{x},\mathbf{w}) = \arg\max_{\mathbf{z}'} \mathbf{w}^T \phi(\mathbf{x},\mathbf{z}') \tag{5}$$

In this case, we identify the vector $\mathbf{v}$ as $\mathbf{v} = \overline{\phi(\mathbf{x},\mathbf{z})} - \phi(\mathbf{x}^*,\mathbf{z}^*)$. In the left-hand term of this expression, $\mathbf{x}$ is clamped to the data-cases and $\mathbf{z}$ is found as in Eqn. 5 by maximizing every data-case separately; in the right-hand (or negative) term, $\mathbf{x}^*, \mathbf{z}^*$ are found by jointly maximizing $\mathbf{w}^T\phi(\mathbf{x},\mathbf{z})$. The quantity $\overline{\phi(\mathbf{x},\mathbf{z})}$ denotes a sample average over the training cases. We note that $\phi(\mathbf{x},\mathbf{z})$ indeed maps to a finite domain because it depends on the real parameter $\mathbf{w}$ only through the discrete state $\mathbf{z}$. We also notice again that $\mathbf{w}^T\mathbf{v} \leq 0$ because of the definition of $(\mathbf{x}^*,\mathbf{z}^*)$. From the convergence theorem we find that, $||\frac{1}{T}\sum_{t=1}^{T} \overline{\phi(\mathbf{x},\mathbf{z}_t)} - \frac{1}{T}\sum_{t=1}^{T}\phi(\mathbf{x}_t^*,\mathbf{z}_t^*)|| \sim \mathcal{O}(1/T)$. This result can be extended to mini-batches as well.

## 2.4 Conditional Herding

We are now ready to propose our new algorithm: *conditional herding* (CH). Like the VP algorithm, CH is concerned with discriminative learning and, therefore, it conditions on the input attributes $\{\mathbf{x}_i\}$. CH differs from VP in that it uses hidden variables, similar to the herder described in the previous subsection. In the most general setting, CH uses features:

$$\phi(\mathbf{x},\mathbf{y},\mathbf{z}), \qquad \mathbf{z}(\mathbf{x},\mathbf{y},\mathbf{w}) = \arg\max_{\mathbf{z}'} \mathbf{w}^T\phi(\mathbf{x},\mathbf{y},\mathbf{z}'). \tag{6}$$

In the experiments in Section 3, we use the explicit form:

$$\mathbf{w}^T\phi(\mathbf{x},\mathbf{y},\mathbf{z}) = \mathbf{x}^T\mathbf{W}\mathbf{z} + \mathbf{y}^T\mathbf{B}\mathbf{z} + \boldsymbol{\theta}^T\mathbf{z} + \boldsymbol{\alpha}^T\mathbf{y}. \tag{7}$$

where $\mathbf{W}$, $\mathbf{B}$, $\boldsymbol{\theta}$ and $\boldsymbol{\alpha}$ are the weights, $\mathbf{z}$ is a binary vector and $\mathbf{y}$ is a binary vector in a 1-of-$K$ scheme (see Figure 1). At each iteration $t$, CH randomly samples a subset of the data-cases and their labels $\mathcal{D}_t = \{\mathbf{x}_{i_t}, \mathbf{y}_{i_t}\} \subseteq \mathcal{D}$. For every member of this mini-batch it computes a hidden variable $\mathbf{z}_{i_t}$ using Eqn. 6. The parameters are then updated as:

$$\mathbf{w}_{t+1} = \mathbf{w}_t + \frac{\eta}{|\mathcal{D}_t|}\sum_{i_t \in \mathcal{D}_t}\left(\phi(\mathbf{x}_{i_t},\mathbf{y}_{i_t},\mathbf{z}_{i_t}) - \phi(\mathbf{x}_{i_t},\mathbf{y}_{i_t}^*,\mathbf{z}_{i_t}^*)\right) \tag{8}$$

In the positive term, $\mathbf{z}_{i_t}$, is found as in Eqn. 5. The negative term is obtained (similar to the perceptron) by making a prediction for the labels, keeping the input attributes fixed:

$$(\mathbf{y}_{i_t}^*, \mathbf{z}_{i_t}^*) = \arg\max_{\mathbf{y}',\mathbf{z}'}\mathbf{w}^T\phi(\mathbf{x}_{i_t},\mathbf{y}',\mathbf{z}'), \quad \forall i_t \in \mathcal{D}_t. \tag{9}$$

For the PCT to apply to CH, the set $\mathbf{V}$ of update vectors must be finite. The inputs $\mathbf{x}$ can be real-valued because we condition on the inputs and there will be at most $N$ distinct values (one for each data-case). However, since we maximize over $\mathbf{y}$ and $\mathbf{z}$ these states must be discrete for the PCT to apply.

Eqn. 8 includes a potentially vector-valued stepsize $\boldsymbol{\eta}$. Notice however that scaling $\mathbf{w} \leftarrow \lambda\mathbf{w}$ will have no affect on the values of $\mathbf{z}, \mathbf{z}^*$ or $\mathbf{y}^*$ and hence on $\mathbf{v}$. Therefore, if we also scale $\boldsymbol{\eta} \leftarrow \lambda\boldsymbol{\eta}$, then the *sequence* of discrete states $\mathbf{z}_t, \mathbf{z}_t^*, \mathbf{y}_t^*$ will not be affected either. Since $\mathbf{w}_t = \boldsymbol{\eta}\sum_{t'=0}^{t-1}\mathbf{v}_{t'} + \mathbf{w}_0$, the only scale that matters is the relative scale between $\mathbf{w}_0$ and $\boldsymbol{\eta}$. In case there would just be a single attractor set for the dynamics of $\mathbf{w}$, the initialization $\mathbf{w}_0$ would only represent a transient affect. However, in practice the scale of $\mathbf{w}_0$ relative to that of $\boldsymbol{\eta}$ does play an important role indicating that many different attractor sets exist for this system.

Irrespective of the attractor we end up in, the PCT guarantees that:

$$||\frac{1}{T}\sum_{t=1}^{T}\frac{1}{|\mathcal{D}_t|}\sum_{i_t}\phi(\mathbf{x}_{i_t},\mathbf{y}_{i_t},\mathbf{z}_{i_t}) - \frac{1}{T}\sum_{t=1}^{T}\frac{1}{|\mathcal{D}_t|}\sum_{i_t}\phi(\mathbf{x}_{i_t},\mathbf{y}_{i_t}^*,\mathbf{z}_{i_t}^*)|| \sim \mathcal{O}(1/T). \tag{10}$$

In general, herding systems perform better when we use normalized features: $\|\phi(\mathbf{x},\mathbf{z},\mathbf{y})\| = R, \; \forall(\mathbf{x},\mathbf{z},\mathbf{y})$. The reason is that herding selects states by maximizing the inner product $\mathbf{w}^T\phi$

and features with large norms will therefore become more likely to be selected. In fact, one can show that states inside the convex hull of the $\phi(\mathbf{x}, \mathbf{y}, \mathbf{z})$ are never selected. For binary ($\pm 1$) variables all states live on the convex hull, but this need not be true in general, especially when we use continuous attributes $\mathbf{x}$. To remedy this, one can either normalize features or add one additional feature[2] $\phi_0(\mathbf{x}, \mathbf{y}, \mathbf{z}) = \sqrt{R_{\max}^2 - ||\phi(\mathbf{x}, \mathbf{y}, \mathbf{z})||^2}$, where $R_{\max} = \max_{\mathbf{x}, \mathbf{y}, \mathbf{z}} \phi(\mathbf{x}, \mathbf{y}, \mathbf{z})$ where $\mathbf{x}$ is only allowed to vary over the data-cases.

Finally, predictions on unseen test data are made by:

$$(\mathbf{y}_{\text{tst},t}^*, \mathbf{z}_{\text{tst},t}^*) = \arg\max_{\mathbf{y}', \mathbf{z}'} \mathbf{w}_t^T \phi(\mathbf{x}_{\text{tst}}, \mathbf{y}', \mathbf{z}'), \tag{11}$$

The algorithm is summarized in the algorithm-box below.

---

Conditional Herding (CH)

1. Initialize $\mathbf{w}_0$ (with finite norm) and $\mathbf{y}_{\text{avg},j} = \mathbf{0}$ for all test cases $j$.
2. For $t \geq 0$:
    (a) Choose a subset $\{\mathbf{x}_{i_t}, \mathbf{y}_{i_t}\} = \mathcal{D}_t \subseteq \mathcal{D}$. For each $(\mathbf{x}_{i_t}, \mathbf{y}_{i_t})$, choose a hidden state $\mathbf{z}_{i_t}$.
    (b) Choose a set of "negative states" $\{(\mathbf{x}_{i_t}^* = \mathbf{x}_{i_t}, \mathbf{y}_{i_t}^*, \mathbf{z}_{i_t}^*)\}$, such that:

$$\frac{1}{|\mathcal{D}_t|} \sum_{i_t} \mathbf{w}_{t-1}^T \phi(\mathbf{x}_{i_t}, \mathbf{y}_{i_t}, \mathbf{z}_{i_t}) \leq \frac{1}{|\mathcal{D}_t|} \sum_{i_t} \mathbf{w}_{t-1}^T \phi(\mathbf{x}_{i_t}, \mathbf{y}_{i_t}^*, \mathbf{z}_{i_t}^*). \tag{12}$$

3. Update $\mathbf{w}_t$ according to Eqn. 8.
4. Predict on test data as follows:
    (a) For every test case $\mathbf{x}_{\text{tst},j}$ at every iteration, choose negative states $(\mathbf{y}_{\text{tst},jt}^*, \mathbf{z}_{\text{tst},jt}^*)$ in the same way as for training data.
    (b) Update online average over predictions, $\mathbf{y}_{\text{avg},j}$, for all test cases $j$.

---

## 2.5 Zero Temperature Limit of Discriminative MRF Learning

Regular herding can be understood as gradient descent on the zero temperature limit of an MRF model. In this limit, gradient updates with constant step size never lead to convergence, irrespective of how small the step size is. Analogously, CH can be viewed as constant step size gradient updates on the zero temperature limit of discriminative MRFs (see [10] for the corresponding RBM model). The finite temperature model is given by:

$$p(\mathbf{y}|\mathbf{x}) = \frac{\sum_{\mathbf{z}} \exp\left[\mathbf{w}^T \phi(\mathbf{y}, \mathbf{z}, \mathbf{x})\right]}{\sum_{\mathbf{z}', \mathbf{y}'} \exp\left[\mathbf{w}^T \phi(\mathbf{y}', \mathbf{z}', \mathbf{x})\right]}. \tag{13}$$

Similar to herding [14], conditional herding introduces a temperature by replacing $\mathbf{w}$ by $\mathbf{w}/T$ and takes the limit $T \to 0$ of $\ell_T \triangleq T\ell$, where $\ell = \sum_i \log p(\mathbf{y}_i | \mathbf{x}_i)$.

# 3 Experiments

We studied the behavior of conditional herding on two artificial and four real-world data sets, comparing its performance to that of the voted perceptron [5] and that of discriminative RBMs [10]. The experiments on artificial and real-world data are discussed separately in Section 3.1 and 3.2.

We studied conditional herding in the discriminative RBM architecture illustrated in Figure 1 (i.e., we use the energy function in Eqn. 7). Per the discussion in Section 2.4, we added an additional feature $\phi_0(\mathbf{x}) = \sqrt{R_{\max}^2 - ||\mathbf{x}||^2}$ with $R_{\max} = \max_i ||\mathbf{x}_i||$ in all experiments.

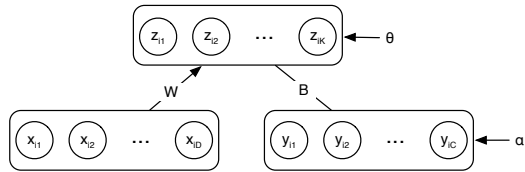

Figure 1: Discriminative Restricted Boltzmann Machine model of distribution $p(\mathbf{y}, \mathbf{z}|\mathbf{x})$.

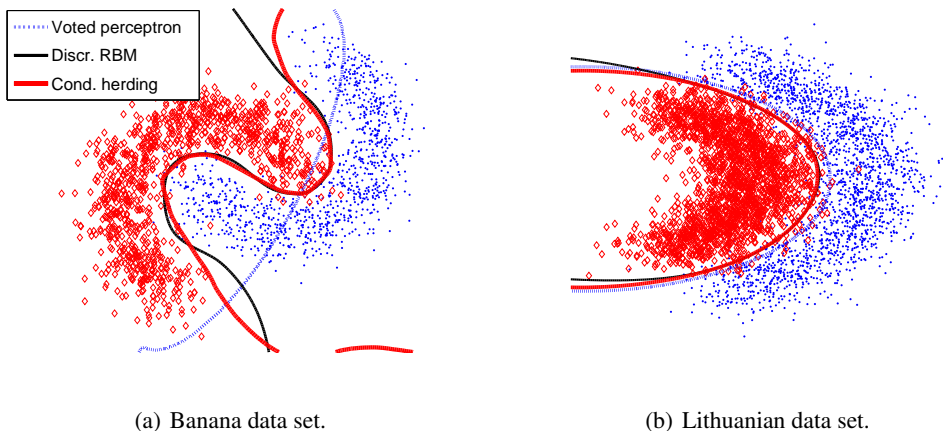

(a) Banana data set.             (b) Lithuanian data set.

Figure 2: Decision boundaries of VP, CH, and dRBMs on two artificial data sets.

## 3.1 Artificial Data

To investigate the characteristics of VP, dRBMs and CH, we used the techniques to construct decision boundaries on two artificial data sets: (1) the banana data set; and (2) the Lithuanian data set. We ran VP and CH for $1,000$ epochs using mini-batches of size $100$. The decision boundary for VP and CH is located at the location where the sign of the prediction $\mathbf{y}_{tst}^*$ changes. We used conditional herders with 20 hidden units. The dRBMs also had 20 hidden units and were trained by running conjugate gradients until convergence. The weights of the dRBMs were initialized by sampling from a Gaussian distribution with a variance of $10^{-4}$. The decision boundary for the dRBMs is located at the point where both class posteriors are equal, i.e., where $p(y_{tst}^* = -1|\tilde{\mathbf{x}}_{tst}) = p(y_{tst}^* = +1|\tilde{\mathbf{x}}_{tst}) = 0.5$.

Plots of the decision boundary for the artificial data sets are shown in Figure 2. The results on the banana data set illustrate the representational advantages of hidden units. Since VP selects data points at random to update the weights, on the banana data set, the weight vector of VP tends to oscillate back and forth yielding a nearly linear decision boundary[3]. This happens because VP can regress on only $2+1=3$ fixed features. In contrast, for CH the simple predictor in the top layer can regress onto $M = 20$ hidden features. This prevents the same oscillatory behavior from occurring.

## 3.2 Real-World Data

In addition to the experiments on synthetic data, we also performed experiments on four real-world data sets - namely, (1) the USPS data set, (2) the MNIST data set, (3) the UCI Pendigits data set, and (4) the 20-Newsgroups data set. The USPS data set consists of 11,000, $16 \times 16$ grayscale images of handwritten digits ($1,100$ images of each digit 0 through 9) with no fixed division. The MNIST data set contains $70,000$, $28 \times 28$ grayscale images of digits, with a fixed division into $60,000$ training and $10,000$ test instances. The UCI Pendigits consists of 16 (integer-valued) features extracted from the movement of a stylus. It contains $10,992$ instances, with a fixed division into $7,494$ training and $3,498$ test instances. The 20-Newsgroups data set contains bag-of-words representations of $18,774$ documents gathered from 20 different newsgroups. Since the bag-of-words representation

comprises over $60,000$ words, we identified the $5,000$ most frequently occurring words. From this set, we created a data set of $4,900$ binary word-presence features by binarizing the word counts and removing the $100$ most frequently occurring words. The 20-Newsgroups data has a fixed division into $11,269$ training and $7,505$ test instances. On all data sets with real-valued input attributes we used the 'normalizing' feature described above.

The data sets used in the experiments are multi-class. We adopted a 1-of-$K$ encoding, where if $\mathbf{y}_i$ is the label for data point $\mathbf{x}_i$, then $\mathbf{y}_i = \{y_{i,1}, ..., y_{i,K}\}$ is a binary vector such that $y_{i,k} = 1$ if the label of the $i^{th}$ data point is $k$ and $y_{i,k} = -1$ otherwise. Performing the maximization in Eqn. 9 is difficult when $K > 2$. We investigated two different procedures for doing so. In the first procedure, we reduce the multi-class problem to a series of binary decision problems using a one-versus-all scheme. The prediction on a test point is taken as the label with the largest online average. In the second procedure, we make predictions on all $K$ labels jointly. To perform the maximization in Eqn. 9, we explore all states of $\mathbf{y}$ in a one-of-$K$ encoding - i.e. one unit is activated and all others are inactive. This partial maximization is not a problem as long as the ensuing configuration satisfies $\mathbf{w}_t^T \mathbf{v}_t \leq 0$ [4]. The main difference between the two procedures is that in the second procedure the weights $\mathbf{W}$ are shared amongst the $K$ classifiers. The primary advantage of the latter procedure is it less computationally demanding than the one-versus-all scheme.

We trained the dRBMs by performing iterations of conjugate gradients (using 3 linesearches) on mini-batches of size $100$ until the error on a small held-out validation set started increasing (i.e., we employed early stopping) or until the negative conditional log-likelihood on the training data stopped coming down. Following [10], we use L2-regularization on the weights of the dRBMs; the regularization parameter was determined based on the generalization error on the same held-out validation set. The weights of the dRBMs were initialized from a Gaussian distribution with variance of $10^{-4}$.

CH used mini-batches of size $100$. For the USPS and Pendigits data sets CH used a burn-in period of $1,000$ updates; on MNIST it was $5,000$ updates; and on 20 Newsgroups it was $20,000$ updates. Herding was stopped when the error on the training set became zero [5].

The parameters of the conditional herders were initialized by sampling from a Gaussian distribution. Ideally, we would like each of the terms in the energy function in Eqn. 7 to contribute equally during updating. However, since the dimension of the data is typically much greater than the number of classes, the dynamics of the conditional herding system will be largely driven by $\mathbf{W}$. To negate this effect, we rescaled the standard deviation of the Gaussian by a factor $1/M$ with $M$ the total number of elements of the parameter involved (e.g. $\sigma_{\mathbf{W}} = \sigma/(\dim(\mathbf{x}) \dim(\mathbf{z}))$ etc.). We also scale the step sizes $\boldsymbol{\eta}$ by the same factor so the updates will retain this scale during herding. The relative scale between $\boldsymbol{\eta}$ and $\sigma$ was chosen by cross-validation. Recall that the absolute scale is unimportant (see Section 2.4 for details).

In addition, during the early stages of herding, we adapted the parameter update for the bias on the hidden units $\boldsymbol{\theta}$ in such a way that the marginal distribution over the hidden units was nearly uniform. This has the advantage that it encourages high entropy in the hidden units, leading to more useful dynamics of the system. In practice, we update $\boldsymbol{\theta}$ as $\boldsymbol{\theta}_{t+1} = \boldsymbol{\theta}_t + \frac{\boldsymbol{\eta}}{|\mathcal{D}_t|} \sum_{i_t} (1 - \lambda) \langle \mathbf{z}_{i_t} \rangle - \mathbf{z}_{i_t}^*$, where $\langle \mathbf{z}_{i_t} \rangle$ is the batch mean. $\lambda$ is initialized to 1 and we gradually half its value every $500$ updates, slowly moving from an entropy-encouraging update to the standard update for the biases of the hidden units.

VP was also run on mini-batches of size $100$ (with step size of 1). VP was run until the predictor started overfitting on a validation set. No burn-in was considered for VP.

The results of our experiments are shown in Table 1. In the table, the best performance on each data set using each procedure is typeset in boldface. The results reveal that the addition of hidden units to the voted perceptron leads to significant improvements in terms of generalization error. Furthermore, the results of our experiments indicate that conditional herding performs on par with discriminative RBMs on the MNIST and USPS data sets and better on the 20 Newsgroups data set. The 20 Newsgroups data is high dimensional and sparse and both VP and CH appear to perform

**One-Versus-All Procedure**

| Technique / Data Set | VP | Discriminative RBM | | Conditional herding | |
|---|---|---|---|---|---|
| | | 100 | 200 | 100 | 200 |
| **MNIST** | 7.69% | **3.57%** | 3.58% | 3.97% | 3.99% |
| **USPS** | 5.03% (0.4%) | 3.97% (0.38%) | 4.02% (0.68%) | 3.49% (0.45%) | **3.35%** (0.48%) |
| **UCI Pendigits** | 10.92% | 5.32% | 5.00% | 3.37% | **3.00%** |
| **20 Newsgroups** | 27.75% | 34.78% | 34.36% | 29.78% | **25.96%** |

**Joint Procedure**

| Technique / Data Set | VP | Discriminative RBM | | | Conditional herding | | |
|---|---|---|---|---|---|---|---|
| | | 50 | 100 | 500 | 50 | 100 | 500 |
| **MNIST** | 8.84% | 3.88% | 2.93% | **1.98%** | 2.89% | 2.09% | 2.09% |
| **USPS** | 4.86% (0.52%) | 3.13% (0.73%) | 2.84% (0.59%) | 4.06% (1.09%) | 3.36% (0.48%) | 3.07% (0.52%) | **2.81%** (0.50%) |
| **UCI Pendigits** | 6.78% | 3.80% | 3.23% | 8.89% | 3.14% | **2.57%** | 2.86% |
| **20 Newsgroups** | **24.89%** | – | 30.57% | 30.07% | – | 25.76% | 24.93% |

Table 1: Generalization errors of VP, dRBMs, and CH on 4 real-world data sets. dRBMs and CH results are shown for various numbers of hidden units. The best performance on each data set is typeset in boldface; missing values are shown as '-'. The std. dev. of the error on the 10-fold cross validation of the USPS data set is reported in parentheses.

quite well in this regime. Techniques to promote sparsity in the hidden layer when training dRBMs exist (see [10]), but we did not investigate them here. It is also worth noting that CH is rather resilient to overfitting. This is particularly evident in the low-dimensional UCI Pendigits data set, where the dRBMs start to badly overfit with 500 hidden units, while the test error for CH remains level. This phenomena is the benefit of averaging over many different predictors.

## 4   Concluding Remarks

The main contribution of this paper is to expose a relationship between the PCT and herding algorithms. This has allowed us to strengthen certain results for herding - namely, theoretically validating herding with mini-batches and partial optimization. It also directly leads to the insight that non-convergent VPs and herding match moments between data and generated predictions at a rate much faster than random sampling ($\mathcal{O}(1/T)$ vs. $\mathcal{O}(1/\sqrt{T})$). From these insights, we have proposed a new conditional herding algorithm that is the zero-temperature limit of dRBMs [10].

The herding perspective provides a new way of looking at learning as a dynamical system. In fact, the PCT precisely specifies the conditions that need to hold for a herding system (in batch mode) to be a piecewise isometry [7]. A piecewise isometry is a weakly chaotic dynamical system that divides parameter space into cells and applies a different isometry in each cell. For herding, the isometry is given by a translation and the cells are labeled by the states $\{\mathbf{x}^*, \mathbf{y}^*, \mathbf{z}, \mathbf{z}^*\}$, whichever combination applies. Therefore, the requirement of the PCT that the space $\mathbf{V}$ must be of finite cardinality translates into the division of parameter space in a finite number of cells, each with its own isometry. Many interesting results about piecewise isometries have been proven in the mathematics literature such as the fact that the sequence of sampled states grows algebraically with $T$ and not exponentially as in systems with random or chaotic components [6]. We envision a fruitful cross-fertilization between the relevant research areas in mathematics and learning theory.

**Acknowledgments**

This work is supported by NSF grants 0447903, 0914783, 0928427 and 1018433 as well as ONR/MURI grant 00014-06-1-073. LvdM acknowledges support by the Netherlands Organisation for Scientific Research (grant no. 680.50.0908) and by EU-FP7 NoE on Social Signal Processing (SSPNet).

## Footnotes

[1]Similar convergence could also be achieved (without concern for generalization performance) by sampling directly from the training data. However, herding converges with rate $1/T$ and is regularized by the weights to prevent overfitting.

[2]If in test data this extra feature becomes imaginary we simply set it to zero.

[3]On the Lithuanian data set, VP constructs a good boundary by exploiting the added 'normalizing' feature.

[4]Local maxima can also be found by iterating over $y_{\text{tst}}^{*,k}, z_{\text{tst},j}^{*,k}$, but the proposed procedure is more efficient.

[5]We use a fixed order of the mini-batches, so that if there are $N$ data cases and the batch size is $K$, if the training error is 0 for $\lceil N/K \rceil$ iterations, the error for the whole training set is 0.

## References

[1] C.M. Bishop. *Pattern Recognition and Machine Learning*. Springer, 2006.

[2] H.D. Block and S.A. Levin. On the boundedness of an iterative procedure for solving a system of linear inequalities. *Proceedings of the American Mathematical Society*, 26(2):229–235, 1970.

[3] Y. Chen and M. Welling. Parametric herding. In *Proceedings of the Thirteenth International Conference on Artificial Intelligence and Statistics*, 2010.

[4] M. Collins. Discriminative training methods for hidden markov models: Theory and experiments with perceptron algorithms. In *Proceedings of the ACL-02 conference on Empirical methods in natural language processing-Volume 10*, page 8. Association for Computational Linguistics, 2002.

[5] Y. Freund and R.E. Schapire. Large margin classification using the perceptron algorithm. *Machine learning*, 37(3):277–296, 1999.

[6] A. Goetz. Perturbations of 8-attractors and births of satellite systems. *Internat. J. Bifur. Chaos, Appl. Sci. Engrg.*, 8(10):1937–1956, 1998.

[7] A. Goetz. Global properties of a family of piecewise isometries. *Ergodic Theory Dynam. Systems*, 29(2):545–568, 2009.

[8] G.E. Hinton. Training products of experts by minimizing contrastive divergence. *Neural Computation*, 14:1771–1800, 2002.

[9] E.T. Jaynes. Information theory and statistical mechanics. *Physical Review Series II*, 106(4):620–663, 1957.

[10] H. Larochelle and Y. Bengio. Classification using discriminative Restricted Boltzmann Machines. In *Proceedings of the $25^{th}$ International Conference on Machine learning*, pages 536–543. ACM, 2008.

[11] M.L. Minsky and S. Papert. *Perceptrons; An introduction to computational geometry*. Cambridge, Mass.,: MIT Press, 1969.

[12] F. Rosenblatt. The perceptron: A probabilistic model for information storage and organization in the brain. *Psychological review*, 65(6):386–408, 1958.

[13] T. Tieleman. Training Restricted Boltzmann Machines using approximations to the likelihood gradient. In *Proceedings of the $25^{th}$ International Conference on Machine learning*, volume 25, pages 1064–1071, 2008.

[14] M. Welling. Herding dynamic weights for partially observed random field models. In *Proc. of the Conf. on Uncertainty in Artificial Intelligence*, Montreal, Quebec, CAN, 2009.

[15] M. Welling. Herding dynamical weights to learn. In *Proceedings of the 21st International Conference on Machine Learning*, Montreal, Quebec, CAN, 2009.

[16] M. Welling and Y. Chen. Statistical inference using weak chaos and infinite memory. In *Proceedings of the Int'l Workshop on Statistical-Mechanical Informatics (IW-SMI 2010)*, pages 185–199, 2010.

[17] L. Younes. Parametric inference for imperfectly observed Gibbsian fields. *Probability Theory and Related Fields*, 82:625–645, 1989.

